# Monte Carlo Methods for Maximum Margin Supervised Topic Models

**Qixia Jiang**[†‡]**, Jun Zhu**[†‡]**, Maosong Sun**[†]**, and Eric P. Xing**[∗∗]
[†]Department of Computer Science & Technology, Tsinghua National TNList Lab,
[†]State Key Lab of Intelligent Tech. & Sys., Tsinghua University, Beijing 100084, China
[∗]School of Computer Science, Carnegie Mellon University, Pittsburgh, PA 15213
{qixia,dcszj,sms}@mail.tsinghua.edu.cn; epxing@cs.cmu.edu

## Abstract

An effective strategy to exploit the supervising side information for discovering predictive topic representations is to impose discriminative constraints induced by such information on the posterior distributions under a topic model. This strategy has been adopted by a number of supervised topic models, such as MedLDA, which employs max-margin posterior constraints. However, unlike the likelihood-based supervised topic models, of which posterior inference can be carried out using the Bayes' rule, the max-margin posterior constraints have made Monte Carlo methods infeasible or at least not directly applicable, thereby limited the choice of inference algorithms to be based on variational approximation with strict mean field assumptions. In this paper, we develop two efficient Monte Carlo methods under much weaker assumptions for max-margin supervised topic models based on an importance sampler and a collapsed Gibbs sampler, respectively, in a convex dual formulation. We report thorough experimental results that compare our approach favorably against existing alternatives in both accuracy and efficiency.

## 1   Introduction

Topic models, such as Latent Dirichlet Allocation (LDA) [3], have shown great promise in discovering latent semantic representations of large collections of text documents. In order to fit data better, LDA has been successfully extended in various ways. One notable extension is supervised topic models, which were developed to incorporate supervising side information for discovering predictive latent topic representations. Representative methods include supervised LDA (sLDA) [2, 12], discriminative LDA (DiscLDA) [8], and max-entropy discrimination LDA (MedLDA) [16].

MedLDA differs from its counterpart supervised topic models by imposing discriminative constraints (i.e., max-margin constraints) directly on the desired posterior distributions, instead of defining a normalized likelihood model as in sLDA and DiscLDA. Such topic models with max-margin posterior constraints have shown superior performance in various settings [16, 14, 13, 9]. However, their constrained formulations, especially when using soft margin constraints for inseparable practical problems, make it infeasible or at least hard if possible at all[1] to directly apply Monte Carlo (MC) methods [10], which have been widely used in the posterior inference of likelihood based models, such as the collapsed Gibbs sampling methods for LDA [5]. Previous inference methods for such models with max-margin posterior constraints have been exclusively on the variational methods [7] usually with a strict mean-field assumption. Although factorized variational methods often seek faster approximation solutions, they could be inaccurate or obtain too compact results [1].

---

[∗‡]indicates equal contributions from these authors.

[1]Rejection sampling can be applied when the constraints are hard, e.g., for separable problems. But it would be inefficient when the sample space is large.

In this paper, we develop efficient Monte Carlo methods for max-margin supervised topic models, which we believe is crucial for highly scalable implementation, and further performance enhancement of this class of models. Specifically, we first provide a new and equivalent formulation of the MedLDA as a regularized Bayesian model with max-margin posterior constraints, based on Zellner's interpretation of Bayes' rule as a learning model [15] and the recent development of regularized Bayesian inference [17]. This interpretation is arguably more natural than the original formulation of MedLDA as a hybrid max-likelihood and max-margin learning, where the log-likelihood is approximated by a variational upper bound for computational tractability. Then, we deal with the set of soft max-margin constraints with convex duality methods and derive the optimal solutions of the desired posterior distributions. To effectively reduce the size of the sampling space, we develop two samplers, namely, an importance sampler and a collapsed Gibbs sampler [4, 1], with a much weaker assumption on the desired posterior distribution compared to the mean field methods in [16]. We note that the work [11] presents a duality method to handle moment matching constraints in maximum entropy models. Our work is an extension of their results to learn topic models, which have nontrivially structured latent variables and also use the general soft margin constraints.

## 2   Latent Dirichlet Allocation

LDA [3] is a hierarchical Bayesian model that posits each document as an admixture of $K$ topics, where each topic $\mathbf{\Phi}_k$ is a multinomial distribution over a $V$-word vocabulary. For document $d$, its topic proportion $\boldsymbol{\theta}_d$ is a multinomial distribution drawn from a Dirichlet prior. Let $\mathbf{w}_d = \{w_{dn}\}_{n=1}^N$ denote the words appearing in document $d$. For the $n$-th word $w_{dn}$, a topic assignment $z_{dn} = k$ is drawn from $\boldsymbol{\theta}_d$ and $w_{dn}$ is drawn from $\mathbf{\Phi}_k$. In short, the generative process of $d$ is

$$\boldsymbol{\theta}_d \sim \mathrm{Dir}(\boldsymbol{\alpha}), \; z_{dn} = k \sim \mathrm{Mult}(\boldsymbol{\theta}_d), \; w_{dn} \sim \mathrm{Mult}(\mathbf{\Phi}_k), \tag{1}$$

where $\mathrm{Dir}(\cdot)$ is a Dirichlet, $\mathrm{Mult}(\cdot)$ is a multinomial. For fully-Bayesian LDA, the topics are also random samples drawn from a Dirichlet prior, i.e., $\mathbf{\Phi}_k \sim \mathrm{Dir}(\boldsymbol{\beta})$.

Let $\mathbf{W} = \{\mathbf{w}_d\}_{d=1}^D$ denote all the words in a corpus with $D$ documents, and define $\mathbf{z}_d = \{z_{dn}\}_{n=1}^N$, $\mathbf{Z} = \{\mathbf{z}_d\}_{d=1}^D$, $\mathbf{\Theta} = \{\boldsymbol{\theta}_d\}_{d=1}^D$. The goal of LDA is to infer the posterior distribution

$$p(\mathbf{\Theta}, \mathbf{Z}, \mathbf{\Phi} | \mathbf{W}, \boldsymbol{\alpha}, \boldsymbol{\beta}) = \frac{p_0(\mathbf{\Theta}, \mathbf{Z}, \mathbf{\Phi} | \boldsymbol{\alpha}, \boldsymbol{\beta}) p(\mathbf{W} | \mathbf{\Theta}, \mathbf{Z}, \mathbf{\Phi})}{p(\mathbf{W} | \boldsymbol{\alpha}, \boldsymbol{\beta})}. \tag{2}$$

Since inferring the true posterior distribution is intractable, researchers must resort to variational [3] or Monte Carlo [5] approximate methods. Although both methods have shown success in various scenarios. They have complementary advantages. For example, variational methods (e.g., mean-field) can be generally more efficient, while MC methods can obtain more accurate estimates.

## 3   MedLDA: a supervised topic model with max-margin constraints

MedLDA extends LDA by integrating the max-margin learning into the procedure of discovering latent topic representations to learn latent representations that are good for predicting class labels or rating scores of a document. Empirically, MedLDA and its various extensions [14, 13, 9] have demonstrated promise in learning more discriminative topic representations. The original MedLDA was designed as a hybrid max likelihood and max-margin learning, where the intractable log-likelihood is approximated by a variational bound. To derive our sampling methods, we present a new interpretation of MedLDA from the perspective of regularized Bayesian inference [17].

### 3.1   Bayesian inference as a learning model

As shown in Eq. (2), Bayesian inference is an information processing rule that projects the prior $p_0$ and empirical evidence to a post-data posterior distribution via the Bayes' rule. It is the core for likelihood-based supervised topic models [2, 12]. A fresh interpretation of Bayesian inference was given by Zellner [15], which leads to our novel interpretation of MedLDA. Specifically, Zellner showed that the posterior distribution by Bayes' rule is the solution of an optimization problem. For instance, the posterior $p(\Theta, \mathbf{Z}, \mathbf{\Phi} | \mathbf{W})$ of LDA is equivalent to the optimum solution of

$$\min_{p(\Theta, \mathbf{Z}, \mathbf{\Phi}) \in \mathcal{P}} \mathrm{KL}[p(\Theta, \mathbf{Z}, \mathbf{\Phi}) \| p_0(\Theta, \mathbf{Z}, \mathbf{\Phi})] - \mathbb{E}_p[\log p(\mathbf{W} | \Theta, \mathbf{Z}, \mathbf{\Phi})], \tag{3}$$

where $\mathrm{KL}(q \| p)$ is the Kullback-Leibler divergence from $q$ to $p$, and $\mathcal{P}$ is the space of probability distributions. We will use $\mathcal{L}(p(\Theta, \mathbf{Z}, \mathbf{\Phi}))$ to denote the objective function.

### 3.2 MedLDA: a regularized Bayesian model

For brevity, we consider the classification model. Let $\mathcal{D} = \{(\mathbf{w}_d, y_d)\}_{d=1}^D$ be a given fully-labeled training set, where the response variable $Y$ takes values from a finite set $\mathcal{Y} = \{1, \ldots, M\}$. MedLDA consists of two parts. The first part is an LDA likelihood model for describing input documents. As in previous work, we use the *partial*[2] likelihood model for $\mathbf{W}$. The second part is a mechanism to consider supervising signal. Since our goal is to discover latent representations $\mathbf{Z}$ that are good for classification, one natural solution is to connect $\mathbf{Z}$ directly to our ultimate goal. MedLDA obtains such a goal by building a classification model on $\mathbf{Z}$. One good candidate of the classification model is the max-margin methods, which avoid defining a normalized likelihood model [12].

Formally, let $\boldsymbol{\eta}$ denote the parameters of the classification model. To make the model fully-Bayesian, we also treat $\boldsymbol{\eta}$ random. Then, we want to infer the joint posterior distribution $p(\boldsymbol{\eta}, \Theta, \mathbf{Z}, \boldsymbol{\Phi}|\mathcal{D})$. For classification, MedLDA defines the following discrimination function

$$F(y, \boldsymbol{\eta}, \mathbf{z}; \mathbf{w}) = \boldsymbol{\eta}^\top \mathbf{f}(y, \bar{\mathbf{z}}), \quad F(y; \mathbf{w}) = \mathbb{E}_{p(\boldsymbol{\eta}, \mathbf{z}|\mathbf{w})}[F(y, \boldsymbol{\eta}, \mathbf{z}; \mathbf{w})], \tag{4}$$

where $\bar{\mathbf{z}}$ is a $K$-dim vector whose element $\bar{z}_k$ equals to $\frac{1}{N}\sum_{n=1}^N \mathbb{I}(z_n = k)$, and $\mathbb{I}(x)$ is an indicator function which equals to 1 when $x$ is true otherwise 0; $\mathbf{f}(y, \bar{\mathbf{z}})$ is an $MK$-dim vector whose elements from $(y-1)K$ to $yK$ are $\bar{\mathbf{z}}$ and all others are zero; and $\boldsymbol{\eta}$ is an $MK$-dimensional vector concatenating $M$ class-specific sub-vectors. With the above definitions, a natural prediction rule is

$$\hat{y} = \operatorname*{argmax}_y F(y; \mathbf{w}), \tag{5}$$

and we would like to "regularize" the properties of the latent topic representations to make them suitable for a classification task. One way to achieve that goal is to take the optimization view of Bayes' theorem and impose the following max-margin constraints to problem (3)

$$F(y_d; \mathbf{w}_d) - F(y; \mathbf{w}_d) \geq \ell_d(y) - \xi_d, \ \forall y \in \mathcal{Y}, \ \forall d, \tag{6}$$

where $\ell_d(y)$ is a non-negative function that penalizes the wrong predictions; $\boldsymbol{\xi} = \{\xi_d\}_{d=1}^D$ are non-negative slack variables for inseparable cases. Let $\mathcal{L}(p) = \mathrm{KL}(p||p_0(\boldsymbol{\eta}, \Theta, \mathbf{Z}, \boldsymbol{\Phi})) - \mathbb{E}_p[\log p(\mathbf{W}|\mathbf{Z}, \boldsymbol{\Phi})]$ and $\Delta\mathbf{f}(y, \bar{\mathbf{z}}_d) = \mathbf{f}(y_d, \bar{\mathbf{z}}_d) - \mathbf{f}(y, \bar{\mathbf{z}}_d)$. Then, we define the soft-margin MedLDA as solving

$$\min_{p(\boldsymbol{\eta}, \Theta, \mathbf{Z}, \boldsymbol{\Phi}) \in \mathcal{P}, \boldsymbol{\xi}} \mathcal{L}(p(\boldsymbol{\eta}, \Theta, \mathbf{Z}, \boldsymbol{\Phi})) + \frac{C}{D}\sum_{d=1}^D \xi_d \tag{7}$$

$$\text{s.t.} : \mathbb{E}_p[\boldsymbol{\eta}^\top \Delta\mathbf{f}(y, \bar{\mathbf{z}}_d)] \geq \ell_d(y) - \xi_d, \ \xi_d \geq 0, \ \forall d, \forall y,$$

where the prior is $p_0(\boldsymbol{\eta}, \Theta, \mathbf{Z}, \boldsymbol{\Phi}) = p_0(\boldsymbol{\eta})p_0(\Theta, \mathbf{Z}, \boldsymbol{\Phi})$.

With the above discussions, we can see that MedLDA is an instance of regularized Bayesian models [17]. Also, problem (7) can be equivalently written as

$$\min_{p(\boldsymbol{\eta}, \Theta, \mathbf{Z}, \boldsymbol{\Phi}) \in \mathcal{P}} \mathcal{L}(p(\boldsymbol{\eta}, \Theta, \mathbf{Z}, \boldsymbol{\Phi})) + C\mathcal{R}(p(\boldsymbol{\eta}, \Theta, \mathbf{Z}, \boldsymbol{\Phi})) \tag{8}$$

where $\mathcal{R} = \frac{1}{D}\sum_d \operatorname{argmax}_y(\ell_d(y) - \mathbb{E}_p[\boldsymbol{\eta}^\top\Delta\mathbf{f}(y, \bar{\mathbf{z}}_d)])$ is the hinge loss, an upper bound of the prediction error on training data.

## 4 Monte Carlo methods for MedLDA

As in other variants of topic models, it is intractable to solve problem (7) or the equivalent problem (8) directly. Previous solutions resort to variational mean-field approximation methods. It is easy to show that the variational EM method in [16] is a coordinate descent algorithm to solve problem (7), with the additional fully-factorized mean-field constraint,

$$p(\boldsymbol{\eta}, \Theta, \mathbf{Z}, \boldsymbol{\Phi}) = p(\boldsymbol{\eta})(\prod_d p(\boldsymbol{\theta}_d)\prod_n p(z_{dn}))\prod_k p(\boldsymbol{\Phi}_k). \tag{9}$$

Below, we present two MC sampling methods to solve the MedLDA problem, with much weaker constraints on $p$, and thus they could be expected to produce more accurate solutions.

Specifically, we assume $p(\boldsymbol{\eta}, \Theta, \mathbf{Z}, \boldsymbol{\Phi}) = p(\boldsymbol{\eta})p(\Theta, \mathbf{Z}, \boldsymbol{\Phi})$. Then, the general procedure is to alternately solve problem (8) by performing the following two steps.

**Estimate** $p(\boldsymbol{\eta})$: Given $p(\boldsymbol{\Theta}, \mathbf{Z}, \boldsymbol{\Phi})$, the subproblem (in an equivalent constrained form) is to solve

$$\min_{p(\boldsymbol{\eta}), \boldsymbol{\xi}} \ \mathrm{KL}(p(\boldsymbol{\eta})\|p_0(\boldsymbol{\eta})) + \frac{C}{D}\sum_{d=1}^{D}\xi_d \tag{10}$$

$$\text{s.t.}: \ \mathbb{E}_p[\boldsymbol{\eta}]^\top \Delta\mathbf{f}(y, \mathbb{E}[\bar{\mathbf{z}}_d]) \geq \ell_d(y) - \xi_d, \ \xi_d \geq 0, \ \forall d, \forall y.$$

By using the Lagrangian methods with multipliers $\boldsymbol{\lambda}$, we have the optimum posterior distribution

$$p(\boldsymbol{\eta}) \propto p_0(\boldsymbol{\eta})e^{\boldsymbol{\eta}^\top \cdot \sum_{d=1}^{D}\sum_y \lambda_d^y \Delta\mathbf{f}(y, \mathbb{E}[\bar{\mathbf{z}}_d])}. \tag{11}$$

For the prior $p_0$, for simplicity, we choose the standard normal prior, i.e., $p_0(\boldsymbol{\eta}) = \mathcal{N}(0, I)$. In this case, $p(\boldsymbol{\eta}) = \mathcal{N}(\boldsymbol{\kappa}, I)$ and the dual problem is

$$\max_{\boldsymbol{\lambda}} \ -\frac{1}{2}\boldsymbol{\kappa}^\top\boldsymbol{\kappa} + \sum_{d=1}^{D}\sum_y \lambda_d^y \ell_d(y) \tag{12}$$

$$\text{s.t.}: \ \sum_y \lambda_d^y \in [0, \frac{C}{D}], \ \forall d.$$

where $\boldsymbol{\kappa} = \sum_{d=1}^{D}\sum_y \lambda_d^y \Delta\mathbf{f}(y, \mathbb{E}[\bar{\mathbf{z}}_d])$. Note that $\boldsymbol{\kappa}$ is the posterior mean of classifier parameters $\boldsymbol{\eta}$, and the element $\kappa_{yk}$ represents the contribution of topic $k$ in classifying a data point to category $y$. This problem is the dual problem of a multi-class SVM [6] and we can solve it (or its primal form) efficiently using existing high-performance SVM learners. We denote the optimum solution of this problem by $(p^*(\boldsymbol{\eta}), \boldsymbol{\kappa}^*, \boldsymbol{\xi}^*, \boldsymbol{\lambda}^*)$.

**Estimate** $p(\boldsymbol{\Theta}, \mathbf{Z}, \boldsymbol{\Phi})$: Given $p(\boldsymbol{\eta})$, the subproblem (in an equivalent constrained form) is to solve

$$\min_{p(\boldsymbol{\Theta}, \mathbf{Z}, \boldsymbol{\Phi}), \boldsymbol{\xi}} \ \mathcal{L}(p(\boldsymbol{\Theta}, \mathbf{Z}, \boldsymbol{\Phi})) + \frac{C}{D}\sum_{d=1}^{D}\xi_d \tag{13}$$

$$\text{s.t.}: \ (\boldsymbol{\kappa}^*)^\top \Delta\mathbf{f}(y, \mathbb{E}_p[\bar{\mathbf{z}}_d]) \geq \ell_d(y) - \xi_d, \ \xi_d \geq 0, \ \forall d, \forall y.$$

Although in theory we can solve this subproblem again using Lagrangian dual methods, it would be hard to derive the dual objective function (if possible at all). Here, we use the same strategy as in [16], that is, to update $p(\boldsymbol{\Theta}, \mathbf{Z}, \boldsymbol{\Phi})$ for only one step with $\boldsymbol{\xi}$ being fixed at $\boldsymbol{\xi}^*$ (the optimum solution of the previous step). It is easy to show that by fixing $\boldsymbol{\xi}$ at $\boldsymbol{\xi}^*$, we will have the optimum solution

$$p(\boldsymbol{\Theta}, \mathbf{Z}, \boldsymbol{\Phi}) \propto p(\mathbf{W}, \mathbf{Z}, \boldsymbol{\Theta}, \boldsymbol{\Phi})e^{(\boldsymbol{\kappa}^*)^\top \sum_{dy}(\lambda_d^y)^* \Delta\mathbf{f}(y, \bar{\mathbf{z}}_d)}, \tag{14}$$

The differences between MedLDA and LDA lie in the above posterior distribution. The first term is the same as the posterior of LDA (the evidence $p(\mathbf{W})$ can be absorbed into the normalization constant). The second term indicates the regularization effects due to the max-margin posterior constraints, which is consistent with our intuition. Specifically, for those data with non-zero Lagrange multipliers (i.e., the data are around the decision boundary or misclassified), the second term will bias the model towards a new posterior distribution that favors more discriminative representations on these "hard" data points.

Now, the remaining problem is how to efficiently draw samples from $p(\boldsymbol{\Theta}, \mathbf{Z}, \boldsymbol{\Phi})$ and estimate the expectations $\mathbb{E}[\bar{\mathbf{z}}]$ as accurate as possible, which are needed in learning classification models. Below, we present two representative samplers – an importance sampler and a collapsed Gibbs sampler.

## 4.1 Importance sampler

To avoid dealing with the intractable normalization constant of $p(\boldsymbol{\Theta}, \mathbf{Z}, \boldsymbol{\Phi})$, one natural choice is to use importance sampling. Importance sampling aims at drawing some samples from a "simple" distribution and the expectation is estimated as a weighted average over these samples. However, directly applying importance sampling to $p(\boldsymbol{\Theta}, \mathbf{Z}, \boldsymbol{\Phi})$ may cause some issues since importance sampling suffers from severe limitations in large sample spaces. Alternatively, since the distribution $p(\boldsymbol{\Theta}, \mathbf{Z}, \boldsymbol{\Phi})$ in Eq. (14) has the factorization form $p(\boldsymbol{\Theta}, \mathbf{Z}, \boldsymbol{\Phi}) = p_0(\boldsymbol{\Theta}, \boldsymbol{\Phi})p(\mathbf{Z}|\boldsymbol{\Theta}, \boldsymbol{\Phi})$, another possible method is to adopt the ancestral sampling strategy to draw sample $(\hat{\boldsymbol{\Theta}}, \hat{\boldsymbol{\Phi}})$ from $p_0(\boldsymbol{\Theta}, \boldsymbol{\Phi})$ and then draw samples from $p(\mathbf{Z}|\hat{\boldsymbol{\Theta}}, \hat{\boldsymbol{\Phi}})$. Although it is easy to draw a sample from the Dirichlet prior $p_0(\boldsymbol{\Theta}, \boldsymbol{\Phi}) = \mathrm{Dir}(\boldsymbol{\alpha})\mathrm{Dir}(\boldsymbol{\beta})$, it would require a large number of samples to get a robust estimate of the expectations $\mathbb{E}[\mathbf{Z}]$. Below, we present one solution to reduce sample space.

One feasible method to reduce the sample space is to collapse $(\boldsymbol{\Theta}, \boldsymbol{\Phi})$ out and directly draw samples from the marginal distribution $p(\mathbf{Z})$. However, this will cause tight couplings between $\mathbf{Z}$ and make the number of samples needed to estimate the expectation grow exponentially with the dimensionality of $\mathbf{Z}$ for importance sampler. A practical sampler for this collapsed distribution would be a Markov chain, as we will present in next section. Here, we propose to use the MAP estimate of $(\boldsymbol{\Theta}, \boldsymbol{\Phi})$ as their "single sample"[3] and proceed to draw samples of $\mathbf{Z}$. Specifically, given $(\hat{\boldsymbol{\Theta}}, \hat{\boldsymbol{\Phi}})$, we have the conditional distribution

$$p(\mathbf{Z}|\hat{\boldsymbol{\Theta}}, \hat{\boldsymbol{\Phi}}) \quad \propto \quad p(\mathbf{W}, \mathbf{Z}|\hat{\boldsymbol{\Theta}}, \hat{\boldsymbol{\Phi}}) e^{(\boldsymbol{\kappa}^*)^\top \sum_{dy} (\lambda_d^y)^* \Delta \mathbf{f}(y, \bar{\mathbf{z}}_d)} = \prod_{d=1}^{D} \prod_{n=1}^{N_d} p(z_{dn}|\hat{\boldsymbol{\theta}}_d, \hat{\boldsymbol{\Phi}}), \quad (15)$$

where
$$p(z_{dn} = k|\hat{\boldsymbol{\theta}}_d, \hat{\boldsymbol{\Phi}}, w_{dn} = t) = \frac{1}{Z_{dn}} \hat{\phi}_{kt} \hat{\theta}_{dk} e^{\frac{1}{N_d} \sum_y (\lambda_d^y)^* (\kappa_{y_d k}^* - \kappa_{yk}^*)} \quad (16)$$

and $Z_{dn}$ is a normalization constant, and $\kappa_{yk}^*$ is the $[(y-1)K + k]$-th element of $\boldsymbol{\kappa}^*$. The difference $(\kappa_{y_d k}^* - \kappa_{yk}^*)$ represents the different contribution of topic $k$ in classifying $d$ to the true category $y_d$ and a wrong category $y$. If the difference is positive, topic $k$ contributes to make a correct prediction for $d$; otherwise, it contributes to make a wrong prediction.

Then, we draw $J$ samples $\{z_{dn}^{(j)}\}_{j=1}^{J}$ from a proposal distribution $g(z)$ and compute the expectations

$$\mathbb{E}[\bar{z}_{dk}] = \frac{1}{N_d} \sum_{n=1}^{N_d} \mathbb{E}[z_{dn}], \forall \bar{z}_{dk} \in \bar{\mathbf{z}}_d \text{ and } \mathbb{E}[z_{dn}] \approx \sum_{j=1}^{J} \frac{\gamma_{dn}^j}{\sum_{j=1}^{J} \gamma_{dn}^j} z_{dn}^{(j)}, \quad (17)$$

where the importance weight $\gamma_{dn}^j$ is

$$\gamma_{dn}^j = \prod_{k=1}^{K} \left( \frac{\hat{\theta}_{dk} \hat{\phi}_{kw_{dn}}}{g(k)} e^{\frac{1}{N_d} \sum_y (\lambda_d^y)^* (\kappa_{y_d k}^* - \kappa_{yk}^*)} \right)^{\mathbb{I}(z_{dn}^{(j)} = k)} \quad (18)$$

With the $J$ samples, we update the MAP estimate $(\hat{\boldsymbol{\Theta}}, \hat{\boldsymbol{\Phi}})$

$$\hat{\theta}_{dk} \propto \frac{1}{J} \sum_{n=1}^{N_d} \sum_{j=1}^{J} \frac{\gamma_{dn}^j}{\sum_{j=1}^{J} \gamma_{dn}^j} \mathbb{I}(z_{dn}^{(j)} = k) + \alpha_k$$
$$\hat{\phi}_{kt} \propto \frac{1}{J} \sum_{d=1}^{D} \sum_{n=1}^{N_d} \sum_{j=1}^{J} \frac{\gamma_{dn}^j}{\sum_{j=1}^{J} \gamma_{dn}^j} \mathbb{I}(z_{dn}^{(j)} = k) \mathbb{I}(w_{dn} = t) + \beta_t. \quad (19)$$

The above two steps are repeated until convergence, initializing $(\hat{\boldsymbol{\Theta}}, \hat{\boldsymbol{\Phi}})$ to be uniform, and the samples from the last iteration are used to estimate the expectation statistics needed in the problem of inferring $p(\boldsymbol{\eta})$.

### 4.2 Collapsed Gibbs sampler

As we have stated, another way to effectively reduce the sample space is to integrate out the intermediate variables $(\boldsymbol{\Theta}, \boldsymbol{\Phi})$ and build a Markov chain whose equilibrium distribution is the resulting marginal distribution $p(\mathbf{Z})$. We propose to use collapsed Gibbs sampling, which has been successfully used for LDA [5]. For MedLDA, we integrate out $(\boldsymbol{\Theta}, \boldsymbol{\Phi})$ and get the marginalized posterior distribution

$$p(\mathbf{Z}) = \frac{p(\mathbf{W}, \mathbf{Z}|\boldsymbol{\alpha}, \boldsymbol{\beta})}{Z_q} e^{(\boldsymbol{\kappa}^*)^\top \sum_d \sum_y (\lambda_d^y)^* \Delta \mathbf{f}(y, \bar{\mathbf{z}}_d)}$$
$$= \frac{1}{Z} \left[ \prod_{d=1}^{D} \frac{\delta(\mathbf{C}_d + \boldsymbol{\alpha})}{\delta(\boldsymbol{\alpha})} e^{(\boldsymbol{\kappa}^*)^\top \sum_y (\lambda_d^y)^* \Delta \mathbf{f}(y, \bar{\mathbf{z}}_d)} \right] \left[ \prod_{k=1}^{K} \frac{\delta(\mathbf{C}_k + \boldsymbol{\beta})}{\delta(\boldsymbol{\beta})} \right], \quad (20)$$

where $\delta(\mathbf{x}) = \frac{\prod_{i=1}^{\dim(\mathbf{x})} \Gamma(x_i)}{\Gamma(\sum_{i=1}^{\dim(\mathbf{x})} x_i)}$, $C_k^t$ is the number of times the term $t$ being assigned to topic $k$ over the whole corpus and $\mathbf{C}_k = \{C_k^t\}_{t=1}^{V}$; $C_d^k$ is the number of times that terms being associated with topic $k$ within the $d$-th document and $\mathbf{C}_d = \{C_d^k\}_{k=1}^{K}$. We can also derive the transition probability of one variable $z_{dn}$ given others which we denote by $\mathbf{Z}_{\neg}$ as:

$$p(z_{dn} = k|\mathbf{Z}_{\neg}, \mathbf{W}_{\neg}, w_{dn} = t) \propto \frac{C_{k,\neg n}^t + \beta_t}{\sum_t C_{k,\neg n}^t + \sum_{t=1}^{V} \beta_t} (C_{d,\neg n}^k + \alpha_k) e^{\frac{1}{N_d} \sum_y (\lambda_d^y)^* (\kappa_{y_d k}^* - \kappa_{yk}^*)} \quad (21)$$

where $C_{\cdot, \neg n}^{\cdot}$ indicates that term $n$ is excluded from the corresponding document or topic.

Again, we can see the difference between MedLDA and LDA (using collapsed Gibbs sampling) from the additional last term in Eq. (21), which is due to the max-margin posterior constraints.

For those data on the margin or misclassified (with non-zero Lagrange multipliers), the last term is non-zero and acts as a regularizer directly affecting the topic assignments of these difficult data.

Then, we use the transition distribution in Eq. (21) to construct a Markov chain. After this Markov chain has converged (i.e., finished the burn-in stage), we draw $J$ samples $\{\mathbf{Z}^{(j)}\}$ and estimate the expectation statistics

$$\mathbb{E}[\bar{z}_{dk}] = \frac{1}{N_d} \sum_{n=1}^{N_d} \mathbb{E}[z_{dn}], \forall \bar{z}_{dk} \in \bar{\mathbf{z}}_d, \text{ and } \mathbb{E}[z_{dn}] = \frac{1}{J} \sum_{j=1}^{J} z_{dn}^{(j)}. \tag{22}$$

## 4.3 Prediction

To make prediction on unlabeled testing data using the prediction rule (5), we take the approach that has been adopted for the variational MedLDA, which uses a point estimate of topics $\mathbf{\Phi}$ from training data and makes prediction based on them. Specifically, we use the MAP estimate $\hat{\mathbf{\Phi}}$ to replace the probability distribution $p(\mathbf{\Phi})$. For the importance sampler, $\hat{\mathbf{\Phi}}$ is computed as in Eq. (19). For the collapsed Gibbs sampler, an estimate of $\hat{\mathbf{\Phi}}$ using the samples is $\hat{\phi}_{kt} \propto \frac{1}{J} \sum_{j=1}^{J} C_k^{t\,(j)} + \beta_t$, where $C_k^{t\,(j)}$ is the times that term $t$ is assigned to topic $k$ in the $j$-th sample.

Given a new document $\mathbf{w}$ to be predicted, for importance sampler, the importance weight should be altered as $\gamma_n^j = \prod_{k=1}^{K} (\theta_k \hat{\phi}_{kw_n} / g(k))^{\mathbb{I}(z_n^{(j)} = k)}$. Then, we approximate the expectation of $\mathbf{z}$ as in Eq. (17). For Gibbs sampler, we infer its latent components $\mathbf{z}$ using the obtained $\hat{\mathbf{\Phi}}$ as $p(z_n = k|\mathbf{z}_{\neg n}) \propto \hat{\phi}_{kw_n}(C_{\neg n}^k + \alpha_k)$, where $C_{\neg n}^k$ is the times that the terms in this document $\mathbf{w}$ assigned to topic $k$ with the $n$-th term excluded. Then, we approximate the $\mathbb{E}[\bar{\mathbf{z}}]$ as in Eq. (22).

# 5 Experiments

We empirically evaluate the importance sampler and the Gibbs sampler for MedLDA (denoted by iMedLDA and gMedLDA respectively) on the 20 Newsgroups data set with a standard list of stop words[4] removed. This data set contains about 20K postings within 20 groups. Due to space limitation, we focus on the multi-class setting.

We use the cutting-plane algorithm [6] to solve the multi-class SVM to infer $p(\boldsymbol{\eta})$ and solve for the lagrange multipliers $\boldsymbol{\lambda}$ in MedLDA. For simplicity, we use the uniform proposal distribution $g$ in iMedLDA. In this case, we can globally draw $J$ (e.g., $= 3 \times K$) samples $\{\mathbf{Z}^{(j)}\}_{j=1}^{J}$ from $g(z)$ outside the iteration loop and only update the importance weights to save time. For gMedLDA, we keep $J$ (e.g., 20) adjacent samples after gMedLDA has converged to estimate the expectation statistics. To be fair, we use the same $C$ for different MedLDA methods. The optimum $C$ is chosen via 5-fold cross validation during the training procedure of fMedLDA from $\{a^2 : a = 1, \ldots, 8\}$. We use symmetric Dirichlet priors for all LDA topic models, i.e., $\boldsymbol{\alpha} = \alpha \mathbf{e}_K$ and $\boldsymbol{\beta} = \beta \mathbf{e}_V$, where $\mathbf{e}_n$ is a $n$-dim vector with every entry being 1. We assess the convergence of a Markov chain when (1) it has run for a maximum number of iterations (e.g., 100), or (2) the relative change in its objective, i.e., $\frac{|L^{t+1} - L^t|}{L^t}$, is less than a tolerance threshold $\epsilon$ (e.g., $\epsilon = 10^{-4}$). We use the same strategy to judge whether the overall inference algorithm converges.

We randomly select 7,505 documents from the whole set as the test set and the rest as the training data. We set the cost parameter $\ell_d(y)$ in problem (7) to be 16, which produces better classification performance than the standard 0/1 cost [16]. To measure the sparsity of the latent representations of documents, we compute the average entropy over test documents: $\frac{1}{|\mathcal{D}_t|} \sum_{d \in \mathcal{D}_t} \mathcal{H}(\boldsymbol{\theta}_d)$. We also measure the sparsity of the inferred topic distributions $\mathbf{\Phi}$ in terms of the average entropy over topics, i.e., $\frac{1}{K} \sum_{k=1}^{K} \mathcal{H}(\mathbf{\Phi}_k)$. All experiments are carried out on a PC with 2.2GHz CPU and 3.6G RAM. We report the mean and standard deviation for each model with 4 times randomly initialized runs.

## 5.1 Performance with different topic numbers

This section compares gMedLDA and iMedLDA with baseline methods. MedLDA was shown to outperform sLDA for document classification. Here, we focus on comparing the performance of MedLDA and LDA when using different inference algorithms. Specifically, we compare with the

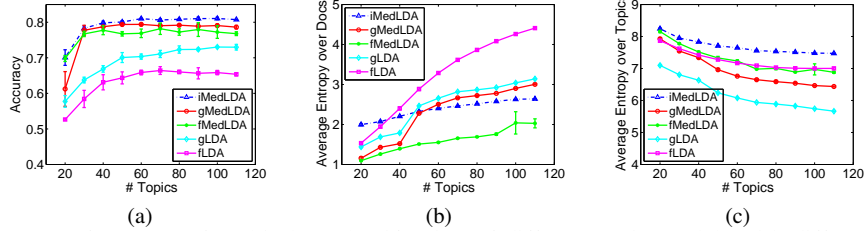

Figure 1: Performance of multi-class classification of different topic models with different topic numbers on 20-Newsgroups data set: (a) classification accuracy, (b) the average entropy of $\Theta$ over test documents, and (c) The average entropy of topic distributions $\Phi$.

LDA model that uses collapsed Gibbs sampling [5] (denoted by gLDA) and the LDA model that uses fully-factorized variational methods [3] (denoted by fLDA). For LDA models, we discover the latent representations of the training documents and use them to build a multi-class SVM classifier. For MedLDA, we report the results when using fully-factorized variational methods (denoted by fMedLDA) as in [16]. Furthermore, fMedLDA and fLDA optimize the hyper-parameter $\alpha$ using the Newton-Rampion method [3], while gMedLDA, iMedLDA and gLDA determine $\alpha$ by 5-fold cross-validation. We have tested a wide range of values of $\beta$ (e.g., $10^{-16} \sim 10^3$) and found that the performance of iMedLDA degrades seriously when $\beta$ is larger than $10^{-3}$. Therefore, we set $\beta$ to be $10^{-5}$ for iMedLDA while $0.01$ for the other topic models just as in the literature [5].

Fig. 1(a) shows the accuracy. We can see that Monte Carlo methods generally outperform the fully-factorized mean-field methods, mainly because of their weaker factorization assumptions. The reason for the superior performance of iMedLDA over gMedLDA is probably because iMedLDA is more effective in dealing with sample sparsity issues. More insights will be provided in Section 5.2.

Fig. 1(b) shows the average entropy of latent representations $\Theta$ over test documents. We find that the entropy of gMedLDA and iMedLDA are smaller than those of gLDA and fLDA, especially for (relatively) large $K$. This implies that sampling methods for MedLDA can effectively concentrate the probability mass on just several topics thus discover more predictive topic representations. However, fMedLDA yields the smallest entropy, which is mainly because the fully-factorized variational methods tend to get too compact results, e.g., sparse local optimums.

Fig. 1(c) shows the average entropy of topic distributions $\Phi$ over topics. We can see that gMedLDA improves the sparsity of $\Phi$ than fMedLDA. However, gMedLDA's entropy is larger than gLDA's. This is because for those "hard" documents, the exponential component in Eq. (21) "regularizes" the conditional probability $p(z_{dn}|\mathbf{Z}_{\neg})$ and leads to a smoother estimate of $\Phi$. On the other hand, we find that iMedLDA has the largest entropy. This is probably because many of the samples (topic assignments) generated by the proposal distribution are "incorrect" but importance sampler still assigns weights to these samples. As a result, the inferred topic distributions are very dense and thus have a large entropy.

Moreover, in the above experiments, we found that the lagrange multipliers in MedLDA are very sparse (about $1\%$ non-zeros for both iMedLDA and gMedLDA; about $1.5\%$ for fMedLDA), much sparser than those of SVM built on raw input data (about $8\%$ non-zeros).

## 5.2 Sensitivity analysis with respect to key parameters

**Sensitivity to $\alpha$.** Fig. 2(a) shows the classification performance of gMedLDA and iMedLDA with different values of $\alpha$. We can see that the performance of gMedLDA increases as $\alpha$ becomes large and retains stable when $\alpha$ is larger than $0.1$. In contrast, the accuracy of iMedLDA decreases a bit (especially for small $K$) when $\alpha$ becomes large, but is relative stable when $\alpha$ is small (e.g., $\leq 0.01$). This is probably because with a finite number of samples, Gibbs sampler tends to produce a too sparse estimate of $\mathbb{E}[\mathbf{Z}]$, and a slightly stronger prior is helpful to deal with the sample sparsity issue. In contrast, the importance sampler avoids such sparsity issue by using a uniform proposal distribution, which could make the samples well cover all topic dimensions. Thus, a small prior is sufficient to get good performance, and increasing the prior's strength could potentially hurt.

**Sensitivity to sample size $J$.** For sampling methods, we always need to decide how many samples (sample size $J$) to keep to ensure sufficient statistics power. Fig. 2(b) shows the classification accuracy of both gMedLDA and iMedLDA with different sample size $J$ when $\alpha = 10^{-2}/K$ and $C = 16$.

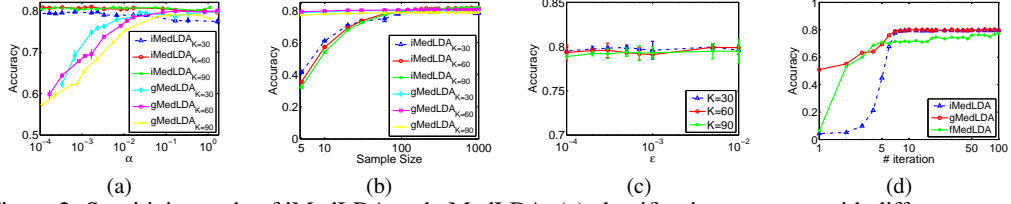

| (a) | (b) | (c) | (d) |

Figure 2: Sensitivity study of iMedLDA and gMedLDA: (a) classification accuracy with different $\alpha$ for different topic numbers, (b) classification accuracy with different sample size $J$, (c) classification accuracy with different convergence criterion $\epsilon$ for gMedLDA, and (d) classification accuracy of different methods varies as a function of iterations when the topic number is 30.

For gMedLDA, we have tested different values of $J$ for training and prediction. We found that the sample size in the training process has almost no influence on the prediction accuracy even when it equals to 1. Hence, for efficiency, we set $J$ to be 1 during the training. It shows that gMedLDA is relatively stable when $J$ is larger than about 20 at prediction. For iMedLDA, Fig. 2(b) shows that it becomes stable when the prediction sample size $J$ is larger than $3 \times K$.

**Sensitivity to convergence criterion $\epsilon$.** For gMedLDA, we have to judge whether a Markov chain has reached its stationarity. Relative change in the objective is a commonly used diagnostic to justify the convergence. We study the influence of $\epsilon$. In this experiment, we don't bound the maximum number of iterations and allow the Gibbs sampler to run until the tolerance $\epsilon$ is reached. Fig. 2(c) shows the accuracy of gMedLDA with different values of $\epsilon$. We can see that gMedLDA is relatively insensitive to $\epsilon$. This is mainly because gMedLDA alternately updates posterior distribution and Lagrangian multipliers. Thus, it does Gibbs sampling for many times, which compensates for the influence that each Markov chain has not reached its stationarity. On the other hand, small $\epsilon$ values can greatly slow the convergence. For instance, when the topic number is 90, gMedLDA takes 11,986 seconds on training when $\epsilon = 10^{-4}$ but 1,795 seconds when $\epsilon = 10^{-2}$. These results imply that we can loose the convergence criterion to speedup training while still obtain a good model.

**Sensitivity to iteration.** Fig. 2(d) shows the the classification accuracy of MedLDA with various inference methods as a function of iteration when the topic number is set at 30. We can see that all the various MedLDA models converge quite quickly to get good accuracy. Compared to fMedLDA, which uses mean-field variational inference, the two MedLDA models using Monte Carlo methods (i.e., iMedLDA and gMedLDA) are slightly faster to get stable prediction performance.

## 5.3 Time efficiency

Although gMedLDA can get good results even for a loosen convergence criterion $\epsilon$ as discussed in Sec. 5.2, we set $\epsilon$ to be $10^{-4}$ for all the methods in order to get a more objective comparison. Fig. 3 reports the total training time of different models, which includes two phases – inferring the latent topic representations and training SVMs. We find iMedLDA is the most efficient, which benefits from (1) generateing samples outside the iteration loop and uses them for

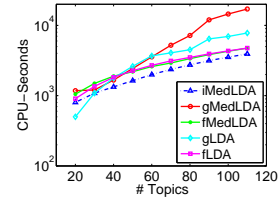

Figure 3: Training time.

all iterations; and (2) using the MAP estimates to collapse the sample space of $(\mathbf{\Theta}, \mathbf{\Phi})$ to a "single sample" for efficiency. In contrast, both gMedLDA and fMedLDA have to iteratively update the variables or variational parameters. gMedLDA requires more time than fMedLDA but is comparable when $\epsilon$ is set to be 0.01. By using the equivalent 1-slack formulation, about $76\%$ of the training time spent on inference for iMedLDA and $90\%$ for gMedLDA. For prediction, both iMedLDA and gMedLDA are slightly slower than fMedLDA.

## 6 Conclusions

We have presented two Monte Carlo methods for MedLDA, a supervised topic model using max-margin constraints directly on the desired posterior distributions for discovering predictive latent topic representations. Our methods are based on a novel interpretation of MedLDA as a regularized Bayesian model and the a convex dual formulation to deal with soft-margin constraints. Experimental results on the 20 Newsgroups data set show that Monte Carlo methods are robust to hyper-parameters and could yield very competitive results for such max-margin topic models.

## Acknowledgements

Part of the work was done when QJ was visiting CMU. JZ and MS are supported by the National Basic Research Program of China (No. 2013CB329403 and 2012CB316301), National Natural Science Foundation of China (No. 91120011, 61273023 and 61170196) and Tsinghua Initiative Scientific Research Program No.20121088071. EX is supported by AFOSR FA95501010247, ONR N000140910758, NSF Career DBI-0546594 and Alfred P. Sloan Research Fellowship.

## Footnotes

[2]A full likelihood model on both $\mathbf{W}$ and $Y$ can be defined as in [12]. But its normalization constant (a function of $\mathbf{Z}$) could make the problem hard to solve.

[3]This collapses the sample space of $(\boldsymbol{\Theta}, \boldsymbol{\Phi})$ to a single point.

[4]http://mallet.cs.umass.edu/

## References

[1] C.M. Bishop. *Pattern recognition and machine learning*, volume 4. springer New York, 2006.

[2] D.M. Blei and J.D. McAuliffe. Supervised topic models. *NIPS*, pages 121–128, 2007.

[3] D.M. Blei, A.Y. Ng, and M.I. Jordan. Latent Dirichlet allocation. *JMLR*, 3:993–1022, 2003.

[4] A. Gelman, J.B. Carlin, H.S. Stern, and D.B. Rubin. *Bayesian data analysis*. Boca Raton, FL: Chapman and Hall/CRC, 2004.

[5] T.L. Griffiths and M. Steyvers. Finding scientific topics. *Proc. of National Academy of Sci.*, pages 5228–5235, 2004.

[6] T. Joachims, T. Finley, and C.N.J. Yu. Cutting-plane training of structural SVMs. *Machine Learning*, 77(1):27–59, 2009.

[7] M.I. Jordan, Z. Ghahramani, T.S. Jaakkola, and L.K. Saul. An introduction to variational methods for graphical models. *Machine learning*, 37(2):183–233, 1999.

[8] S. Lacoste-Jullien, F. Sha, and M.I. Jordan. DiscLDA: Discriminative learning for dimensionality reduction and classification. *NIPS*, pages 897–904, 2009.

[9] D. Li, S. Somasundaran, and A. Chakraborty. A combination of topic models with max-margin learning for relation detection. In *ACL TextGraphs-6 Workshop*, 2011.

[10] R.Y. Rubinstein and D.P. Kroese. *Simulation and the Monte Carlo method*, volume 707. Wiley-interscience, 2008.

[11] E. Schofield. *Fitting maximum-entropy models on large sample spaces*. PhD thesis, Department of Computing, Imperial College London, 2006.

[12] C. Wang, D.M. Blei, and Li F.F. Simultaneous image classification and annotation. *CVPR*, 2009.

[13] Y. Wang and G. Mori. Max-margin latent Dirichlet allocation for image classification and annotation. In *BMVC*, 2011.

[14] S. Yang, J. Bian, and H. Zha. Hybrid generative/discriminative learning for automatic image annotation. In *UAI*, 2010.

[15] A. Zellner. Optimal information processing and Bayes's theorem. *American Statistician*, pages 278–280, 1988.

[16] J. Zhu, A. Ahmed, and E.P. Xing. MedLDA: maximum margin supervised topic models for regression and classification. In *ICML*, pages 1257–1264, 2009.

[17] J. Zhu, N. Chen, and E.P. Xing. Infinite latent SVM for classification and multi-task learning. In *NIPS*, 2011.

